# Comparing the prediction accuracy of artificial neural networks and other statistical models for breast cancer survival

**Harry B. Burke**
Department of Medicine
New York Medical College
Valhalla, NY 10595

**David B. Rosen**
Department of Medicine
New York Medical College
Valhalla, NY 10595

**Philip H. Goodman**
Department of Medicine
University of Nevada School of Medicine
Reno, Nevada 89520

## Abstract

The TNM staging system has been used since the early 1960's to predict breast cancer patient outcome. In an attempt to increase prognostic accuracy, many putative prognostic factors have been identified. Because the TNM stage model can not accommodate these new factors, the proliferation of factors in breast cancer has lead to clinical confusion. What is required is a new computerized prognostic system that can test putative prognostic factors and integrate the predictive factors with the TNM variables in order to increase prognostic accuracy. Using the area under the curve of the receiver operating characteristic, we compare the accuracy of the following predictive models in terms of five year breast cancer-specific survival: pTNM staging system, principal component analysis, classification and regression trees, logistic regression, cascade correlation neural network, conjugate gradient descent neural, probabilistic neural network, and backpropagation neural network. Several statistical models are significantly more ac-

curate than the TNM staging system. Logistic regression and the backpropagation neural network are the most accurate prediction models for predicting five year breast cancer-specific survival

# 1  INTRODUCTION

For over thirty years measuring cancer outcome has been based on the TNM staging system (tumor size, number of lymph nodes with metastatic disease, and distant metastases) (Beahr et. al., 1992). There are several problems with this model (Burke and Henson, 1993). First, it is not very accurate, for breast cancer it is 44% accurate. Second its accuracy can not be improved because predictive variables can not be added to the model. Third, it does not apply to all cancers. In this paper we compare computerized prediction models to determine if they can improve prognostic accuracy. Artificial neural networks (ANN) are a class of non-linear regression and discrimination models. ANNs are being used in many areas of medicine, with several hundred articles published in the last year. Representative areas of research include anesthesiology (Westenskow et. al., 1992), radiology (Tourassi et. al., 1992), cardiology (Leong and Jabri, 1982), psychiatry (Palombo, 1992), and neurology (Gabor and Seyal, 1992). ANNs are being used in cancer research including image processing (Goldberg et. al., 1992) , analysis of laboratory data for breast cancer diagnosis (O Leary et. al., 1992), and the discovery of chemotherapeutic agents (Weinstein et. al., 1992). It should be pointed out that the analyses in this paper rely upon previously collected prognostic factors. These factors were selected for collection because they were significant in a generalized linear model such as the linear or logistic models. There is no predictive model that can improve upon linear or logistic prediction models when the predictor variables meet the assumptions of these models and there are no interactions. Therefore he objective of this paper is not to outperform linear or logistic models on these data. Rather, our objective is to show that, with variables selected by generalized linear models, artificial neural networks can perform as well as the best traditional models . There is no a priori reason to believe that future prognostic factors will be binary or linear, and that there will not be complex interactions between prognostic factors. A further objective of this paper is to demonstrate that artificial neural networks are likely to outperform the conventional models when there are unanticipated nonmonotonic factors or complex interactions.

# 2  METHODS

## 2.1  DATA

The Patient Care Evaluation (PCE) data set is collected by the Commission on Cancer of the American College of Surgeons (ACS). The ACS, in October 1992, requested cancer information from hospital tumor registries in the United States. The ACS asked for the first 25 cases of breast cancer seen at that institution in 1983, and it asked for follow up information on each of these 25 patients through the date of the request. These are only cases of first breast cancer. Follow-up information included known deaths. The PCE data set contains, at best, eight year follow-up.

We chose to use a five year survival end-point. This analysis is for death due to breast cancer, not all cause mortality.

For this analysis cases with missing data, and cases censored before five years, are not included so that the prediction models can be compared without putting any prediction model at a disadvantage. We randomly divided the data set into training, hold-out, and testing subsets of 3,100, 2,069, and 3,102 cases, respectively.

## 2.2  MODELS

The TMN stage model used in this analysis is the pathologic model (pTNM) based on the 1992 American Joint Committee on Cancer's Manual for the Staging of Cancer (Beahr et. al., 1992). The pathologic model relies upon pathologically determined tumor size and lymph nodes, this contrasts with clinical staging which relies upon the clinical examination to provide tumor size and lymph node information. To determine the overall accuracy of the TNM stage model we compared the model's prediction for each patient, where the individual patient's prediction is the fraction of all the patients in that stage who survive, to each patient's true outcome.

Principal components analysis, is a data reduction technique based on the linear combinations of predictor variables that minimizes the variance across patients (Jollie, 1982). The logistic regression analysis is performed in a stepwise manner, without interaction terms, using the statistical language S-PLUS (S-PLUS, 1992), with the continuous variable age modeled with a restricted cubic spline to avoid assuming linearity (Harrell et. al., 1988). Two types of Classification and Regression Tree (CART) (Breiman et. al., 1984) analyses are performed using S-PLUS. The first was a 9-node pruned tree (with 10-fold cross validation on the deviance), and the second was a shrunk tree with 13.7 effective nodes.

The multilayer perceptron neural network training in this paper is based on the maximum likelihood function unless otherwise stated, and backpropagation refers to gradient descent. Two neural networks that are not multilayer perceptrons are tested. They are the Fuzzy ARTMAP neural network (Carpenter et. al., 1991) and the probabilistic neural network (Specht, 1990).

## 2.3  ACCURACY

The measure of comparative accuracy is the area under the curve of the receiver operating characteristic (Az). Generally, the Az is a nonparametric measure of discrimination. Square error summarizes how close each patient's predicted value is to its true outcome. The Az measures the relative goodness of the set of predictions as a whole by comparing the predicted probability of each patient with that of all other patients. The computational approach to the Az that employs the trapezoidal approximation to the area under the receiver operating characteristic curve for binary outcomes was first reported by Bamber (Bamber, 1975), and later in the medical literature by Hanley (Hanley and McNeil, 1982). This was extended by Harrell (Harrell et. al., 1988) to continuous outcomes.

Table 1: PCE 1983 Breast Cancer Data: 5 Year Survival Prediction, 54 Variables.

| PREDICTION MODEL | ACCURACY* | SPECIFICATIONS |
|---|---|---|
| pTNM Stages | .720 | 0,I,IIA,IIB,IIIA,IIIB,IV |
| Principal Components Analysis | .714 | one scaling iteration |
| CART, pruned | .753 | 9 nodes |
| CART, shrunk | .762 | 13.7 nodes |
| Stepwise Logistic regression | .776 | with cubic splines |
| Fuzzy ARTMAP ANN | .738 | 54-F2a, 128-1 |
| Cascade correlation ANN | .761 | 54-21-1 |
| Conjugate gradient descent ANN | .774 | 54-30-1 |
| Probabilistic ANN | .777 | bandwidth = 16s |
| Backpropagation ANN | .784 | 54-5-1 |

\* The area under the curve of the receiver operating characteristic.

# 3  RESULTS

All results are based on the independent variable sample not used for training (i.e., the testing data set), and all analyses employ the same testing data set. Using the PCE breast cancer data set, we can assess the accuracy of several prediction models using the most powerful of the predictor variables available in the data set (See Table 1).

Principal components analysis is not expected to be a very accurate model; with one scaling iteration, its accuracy is .714. Two types of classification and regression trees (CART), pruned and shrunk, demonstrate accuracies of .753 and .762, respectively. Logistic regression with cubic splines for age has an accuracy of .776. In addition to the backpropagation neural network and the probabilistic neural network, three types of neural networks are tested. Fuzzy ARTMAP's accuracy is the poorest at .738. It was too computationally intensive to be a practical model. Cascade-correlation and conjugate gradient descent have the potential to do as well as backpropagation. The PNN accuracy is .777. The PNN has many interesting features, but it also has several drawbacks including its storage requirements. The backpropagation neural network's accuracy is .784.4.

# 4  DISCUSSION

For predicting five year breast cancer-specific survival, several computerized prediction models are more accurate than the TNM stage system, and artificial neural networks are as good as the best traditional statistical models.

## References

Bamber D (1975). The area above the ordinal dominance graph and the area below the receiver operating characteristic. *J Math Psych* 12:387-415.

Beahrs OH, Henson DE, Hutter RVP, Kennedy BJ (1992). *Manual for staging of*

*cancer*, 4th ed. Philadelphia: JB Lippincott.

Burke HB, Henson DE (1993). Criteria for prognostic factors and for an enhanced prognostic system. *Cancer* 72:3131-5.

Breiman L, Friedman JH, Olshen RA (1984). *Classification and Regression Trees.* Pacific Grove, CA: Wadsworth and Brooks/Cole.

Carpenter GA, Grossberg S, Rosen DB (1991). Fuzzy ART: Fast stable learning and categorization of analog patterns by an adaptive resonance system. *Neural Networks* 4:759-771.

Gabor AJ, M. Seyal M (1992) . Automated interictal EEG spike detection using artificial neural networks. *Electroencephalogr Clin Neurophysiology* 83:271-80.

Goldberg V, Manduca A, Ewert DL (1992). Improvement in specificity of ultra-sonography for diagnosis of breast tumors by means of artificial intelligence. *Med Phys* 19:1275-81.

Hanley JA, McNeil BJ (1982). The meaning of the use of the area under the receiver operating characteristic (ROC) curve. *Radiology* 143:29-36.

Harrell FE, Lee KL, Pollock BG (1988). Regression models in clinical studies: determining relationships between predictors and response. *J Natl Cancer Instit* 80:1198-1202.

Jollife IT (1986). *Principal Component Analysis.* New York: Springer-Verlag, 1986.

Leong PH, Jabri MA (1982). MATIC - an intracardiac tachycardia classification system. *PACE* 15:1317-31, 1982.

O'Leary TJ, Mikel UV, Becker RL (1992). Computer-assisted image interpretation: use of a neural network to differentiate tubular carcinoma from sclerosing adenosis. *Modern Pathol* 5:402-5.

Palombo SR (1992). Connectivity and condensation in dreaming. *J Am Psychoanal Assoc* 40:1139-59.

*S-PLUS* (1991), v 3.0. Seattle, WA; Statistical Sciences, Inc.

Specht DF (1990). Probabilistic neural networks. Neural Networks 3:109-18.

Tourassi GD, Floyd CE, Sostman HD, Coleman RE (1993). Acute pulmonary embolism: artificial neural network approach for diagnosis. *Radiology* 189:555-58.

Weinstein JN, Kohn KW, Grever MR et. al. (1992) Neural computing in cancer drug development: predicting mechanism of action. *Science* 258:447-51.

Westenskow DR, Orr JA, Simon FH (1992). Intelligent alarms reduce anesthesiologist's response time to critical faults. *Anesthesiology* 77:1074-9, 1992.

